# Analyzing and Visualizing Single-Trial Event-Related Potentials

**Tzyy-Ping Jung[1,2], Scott Makeig[2,3], Marissa Westerfield[2]**
**Jeanne Townsend[2], Eric Courchesne[2], Terrence J. Sejnowski[1,2]**

[1]Howard Hughes Medical Institute and Computational Neurobiology Laboratory
The Salk Institute, P.O. Box 85800, San Diego, CA 92186-5800
{jung,scott,terry}@salk.edu
[2]University of California, San Diego, La Jolla, CA 92093
[3]Naval Health Research Center, P.O. Box 85122, San Diego, CA 92186-5122

## Abstract

Event-related potentials (ERPs), are portions of electroencephalographic (EEG) recordings that are both time- and phase-locked to experimental events. ERPs are usually averaged to increase their signal/noise ratio relative to non-phase locked EEG activity, regardless of the fact that response activity in single epochs may vary widely in time course and scalp distribution. This study applies a linear decomposition tool, Independent Component Analysis (ICA) [1], to multichannel single-trial EEG records to derive spatial filters that decompose single-trial EEG epochs into a sum of temporally independent and spatially fixed components arising from distinct or overlapping brain or extra-brain networks. Our results on normal and autistic subjects show that ICA can separate artifactual, stimulus-locked, response-locked, and non-event related background EEG activities into separate components, allowing (1) removal of pervasive artifacts of all types from single-trial EEG records, and (2) identification of both stimulus- and response-locked EEG components. Second, this study proposes a new visualization tool, the 'ERP image', for investigating variability in latencies and amplitudes of event-evoked responses in spontaneous EEG or MEG records. We show that sorting single-trial ERP epochs in order of reaction time and plotting the potentials in 2-D clearly reveals underlying patterns of response variability linked to performance. These analysis and visualization tools appear broadly applicable to electrophyiological research on both normal and clinical populations.

# 1 Introduction

Scalp-recorded *event-related potentials* (ERPs) are voltage changes in the ongoing *electroencephalogram* (EEG) that are both time- and phase-locked to some experimental events. These field potentials are usually averaged to increase their signal/noise ratio relative to artifacts and other non-phase locked EEG activity. The averaging method disregards the fact that in single epochs response activity may vary widely in both time course and scalp distribution. These differences are in part attributed to different strategies employed by subjects for processing different stimuli, to differences in expectation, attention, and arousal occurring in different trials, and/or to variations in alertness and fatigue [2, 3]. Single-trial analysis, on the other hand, can avoid problems due to time and/or phase shifts and can potentially reveal much richer information about event-related brain dynamics in endogenous ERPs, but suffers from pervasive artifacts associated with blinks, eye-movements, and muscle noise, and poor signal-to-noise ratio arising from the fact that non-phase locked background EEG activities often are larger than phase-locked response components.

We present here new methods for analyzing and visualizing multichannel unaveraged single-trial ERP records that alleviate these problems. First, multi-channel EEG epochs were analyzed using Independent Component Analysis (ICA), a signal processing technique that can decompose multichannel complex data into spatially fixed and temporally independent components. Next, a new visualization tool, the 'ERP image', is introduced for visualizing relations between single-trial ERP records and their contributions to the ERP average. To form an ERP image, the recorded potentials at one channel are plotted as parallel lines and single-trial ERP epochs are sorted in order of reaction time. ICA, applied to the single-trial EEG records from normal and autistic subjects in a visual selective attention experiment, derived components whose dynamics were affected by stimulus presentations and/or subject responses in distinct ways. We demonstrate, through analysis of two sample data sets, the power of the proposed analysis and visualization tools for increasing the amount and quality of information about event-related brain dynamics that can be derived from single-trial EEG data.

# 2 Independent Component Analysis of EEG data

Bell and Sejnowski [5] have proposed a simple neural network algorithm that blindly separates mixtures, $\mathbf{x}$, of independent sources, s, using infomax. They showed that maximizing the joint entropy, $H(\mathbf{y})$, of the output of a neural processor minimizes the mutual information among the output components, $y_i = g(u_i)$, where $g(u_i)$ is an invertible bounded nonlinearity and $\mathbf{u} = \mathbf{W}\mathbf{x}$, a version of the original sources, s, identical save for scaling and permutation. Lee et al. [1] generalized the infomax algorithm to perform blind source separation on linear mixtures of sources with either sub- or super-Gaussian distributions. Please see [5, 1] for details regarding the algorithms.

ICA is suitable for performing blind source separation on EEG data because: (1) it is plausible that EEG data recorded at multiple scalp sensors are linear sums of temporally independent components arising from spatially fixed, distinct or overlapping brain or extra-brain networks, and, (2) spatial smearing of EEG data by volume conduction does not involve significant time delays[1]. In single-trial EEG analysis, the rows of the input matrix $\mathbf{x}$ are the EEG signals recorded at different electrodes, while the columns are measurements recorded at different time points.

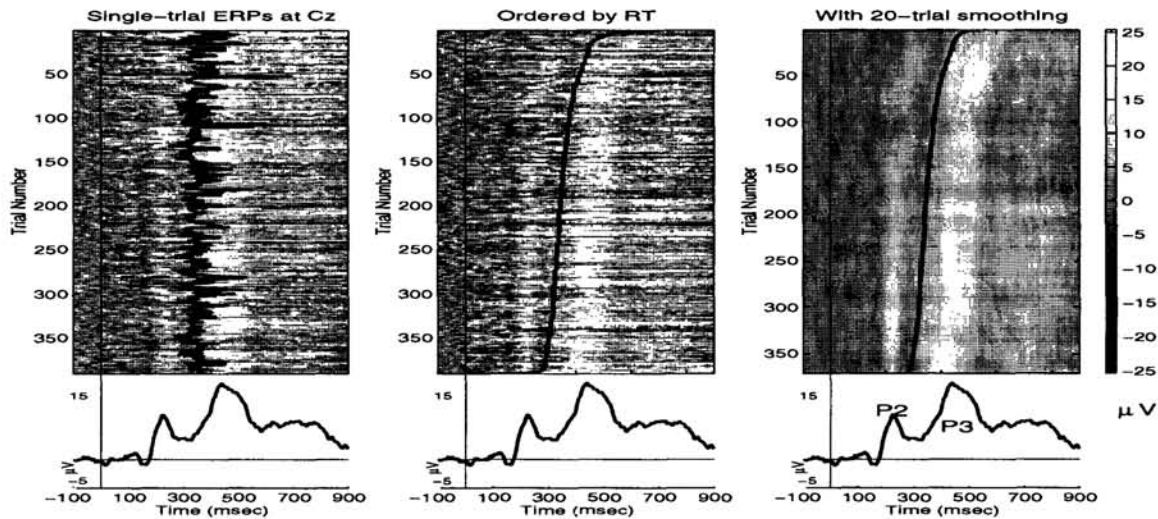

Figure 1: ERP images. *(left panel)* Single-trial ERPs recorded at a central electrode (Cz) and time-locked to onsets of visual target stimuli *(vertical left line)*, plotted with subject reaction times *(thick black line)*. *(middle panel)* The 390 single trials were then sorted (bottom to top) in order of increasing reaction time. *(right panel)* To increase signal-to-noise ratio and minimize EEG signals not both time- and phase-locked to the experimental events, the trials were averaged vertically using a 30-trial moving window advanced in one-trial increments.

The rows of the independent output data matrix $u = Wx$ are time courses of activation of the ICA components, and the columns of the inverse matrix, $W^{-1}$, give the projection strengths of the respective components onto the scalp sensors. The scalp topographies of the components provide evidence as to their physiological origin (e.g., eye activity should project mainly to frontal sites). EEG signals of interest (e.g., event-related brain signals) can then be obtained by projecting selected ICA components back onto the scalp as $x' = (W)^{-1}u'$, where $u'$ is the matrix of activation waveforms, $u$, with rows representing activations of "irrelevant" sources set to zero.

## 3   Methods and Materials

EEG data were recorded at 29 scalp electrodes and 2 EOG placements from 2 normal and 1 autistic subjects who participated in a 2-hr visual selected attention task in which they were instructed to attend to circles flashed in random order at one of five locations laterally arrayed 0.8 cm above a central fixation point. Locations were outlined by five evenly spaced 1.6-cm blue squares displayed on a black background at visual angles of ±2.7 deg and ±5.5 deg from fixation. Attended locations were highlighted through entire 90-sec experimental blocks. Subjects were instructed to maintain fixation on the central cross and press a button each time they saw a circle in the attended location (see [6] for details).

## 4   Results

The ICA algorithm was applied separately to concatenated 31-channel single-trial EEG records from two normal and one autistic subjects. The derived independent components had a variety of distinct relations to task events. Some were clearly time-locked to stimuli presentations, while others were time-locked to subject re-

sponses. Still others captured spontaneous EEG activity together with blinks, eye-movements, and muscle artifacts, while others accounted for oscillatory and other background EEG phenomena.

## 4.1 ERP image

To investigate variability in the latencies and amplitudes of event-evoked responses in spontaneous EEG, we here introduce a new visualization tool, the ERP image. An example shown in Figure 1 (*left panel*) plots 390 single-trial ERP epochs time-locked to onsets of target stimuli (vertical left line) and recorded at a central electrode (Cz) from a normal subject. Each horizontal trace represents a 1-sec single-trial ERP record whose potential variations are plotted in different colors. The thick line plots the subject reaction times (RT) in successive trials. Note the trial-to-trial fluctuations in ERP latency and reaction time. The ERP average of these trials is plotted in the bottom of the panel. Next, the single trials were sorted in order of increasing reaction time (Fig. 1 *middle panel*), and were then smoothed with a 30-trial moving average (*right panel*). Note that, in all but the longest-RT trials, the early positive feature (P2) is time-locked to stimulus onset (i.e. is stimulus-locked), and that the P3 feature follows RT in nearly all trials (i.e. is response-locked). ERP image plots allow visualization of relations between event-related EEG trials and single-trial contributions to their ERP averaged. They disclose a tight link between the amplitudes and latencies of individual event-related responses and subject behavior.

## 4.2 Removing blink and eye-movement artifacts from EEG records

Autistic subjects tend to blink more frequently than normal subjects [8]. ICA, applied to this data set in which about 50% of the trials were contaminated by blinks, successfully isolated blink artifacts to a single component (Fig. 2A, *left*) whose contributions could be removed from the EEG records by subtracting out the component projection [7]. Though the subjects were instructed to fixate during each 90-sec blocks, it has been suspected, though poorly documented, that their eyes tended to drift towards target stimuli presented at peripheral locations. Here, a second ICA component accounted for these small horizontal eye-movements (Fig. 2B, *right*). Fig. 2B (*5 traces*) also shows separate ERP averages (at periocular site EOG2) of responses to targets presented at the five different attended locations. The size of the prominent eye movement-related component is proportional to the angle between the stimulus location and the fixation point. Figure 2C shows the averaged ERPs at the same site in response to stimuli presented at the five different attended locations, before (*faint traces*) and after (*solid traces*) artifact removal. After artifact correction, the averaged ERPs to stimuli presented at the five different locations were independent of stimulus location.

## 4.3 Extracting event-related brain activity from EEG Records

In these data, ICA also separated stimulus-locked, response-locked, and non-phase locked background EEG activities into different independent components. Numbers of components in each class varied across subjects. Figure 3A shows the projections of the subgroups of ICA components accounting primarily for (*left*) stimulus-locked, (*middle*) response-locked, and (*right*) remaining non-phase locked background EEG activity at site PO3. Notice that, (1) both the response latencies and active durations of the early stimulus-locked P1 and N1 components were very stable in nearly all trials, (2) the peak of the later P3 component covaried with reaction time, and (3) the projections of ICA components accounting for non-phase locked background EEG activity contributed very little to the averaged ERP (*right panel*, bottom

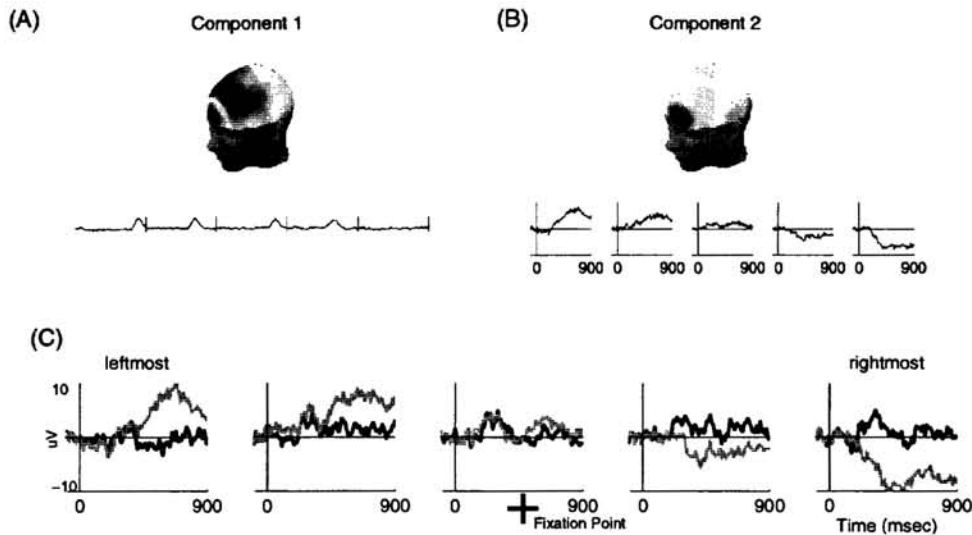

Figure 2: (**A**) (*left*) Scalp topography and 5 consecutive 1-sec epochs of the activation time course of an ICA component counting for blink artifacts in 641 single trials recorded from an adult autistic subject. (**B**) The scalp topography of a second eye-movement component and its averaged activation time courses in response to target stimuli presented at the five different attended locations. (**C**) Averaged ERPs at site EOG2 to targets presented at each of five attended locations, before (*faint traces*) and after (*solid traces*) artifact removal.

trace). These results indicate that ICA makes possible the extraction and separation of event-related brain phenomena of all types from single-trial EEG records.

## 4.4   Re-aligning single-trial event-related potentials

Figure 3B (*left panel*) shows the raw artifact-corrected single-trial ERP epochs (the sum of the data in Fig. 3A). Response latency fluctuations resulted in temporal smearing of the P3 feature in the averaged ERP (*bottom left*). Realigning the single-trial ERP epochs to the median reaction time sharpened the averaged P3 (*center panel*, P3'), but unfortunately made the early stimulus-locked activity out of phase and the early averaged ERP thus absent in the first 200 msec. Because ICA separated stimulus-locked and response-locked activity into different independent components, we could realign the time courses of the response-locked P3' component to the median reaction time and project the adjusted data, along with the unaligned time courses of stimulus-locked components (P1/N1), back onto the scalp sensors (*right panel*). This realignment preserved the early stimulus-locked P1/N1 while sharpening the response-locked P3. The method minimized temporal smearing in the averaged ERP arising from performance fluctuations (*left & right* panels).

## 4.5   Event-related oscillatory EEG activity

ICA, applied to multichannel single-trial EEG records, can also separate multiple oscillatory components even within a single frequency band. For example, Figure 3C plots scalp topographies and ERP images of activations of two ICA components accounting for alpha activity in target-response epochs from a normal subject. Note that the activity of the first component (*left panel*) was augmented following stimulation, while the activity of the second component (*middle panel*) was blocked by the subject response. When the same spatial filter was applied to EEG records from another session in which the subject was instructed to attend to but not to respond

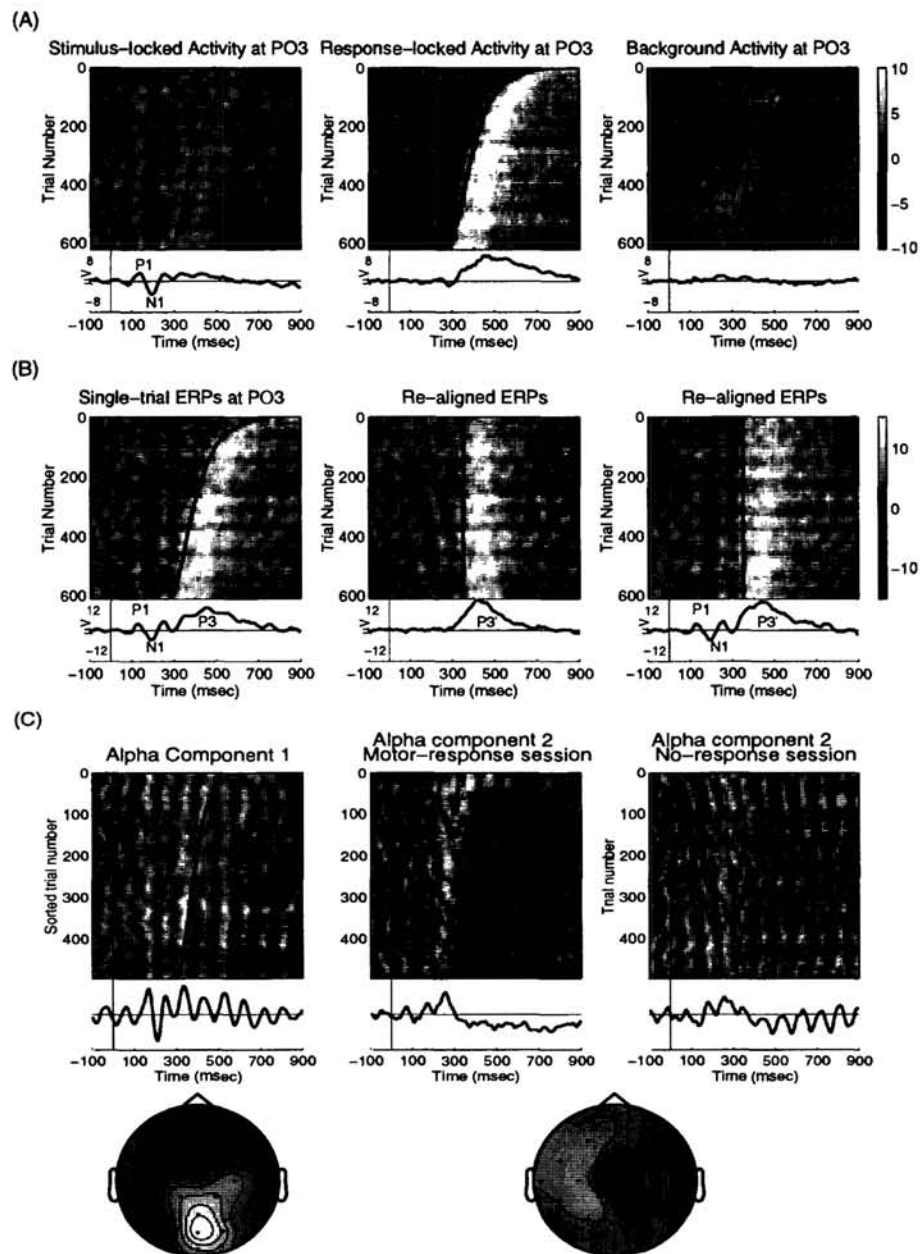

Figure 3: **(A)** Projections of ICA components at site PO3 accounting, respectively, for stimulus-locked (*left*), response-locked (*middle*), and non-phase locked background EEG activity (*right*) at one posterior site, PO3. **(B)** (*left*) Artifact-corrected single-trial ERP records time-locked to stimulus onsets (*left*), and subject responses (*center*). Note that the early ERP features (P1, N1) are not in phase in the response-locked trials, and do not appear in the response-locked average (*center bottom*). (*right*) Projections of the response-locked components were aligned to median reaction time (355 ms) and summed with stimulus-aligned component projections, forming an enhanced stimulus-aligned ERP (right bottom). **(C)** ERP-image plots of activations of ICA components accounting for alpha activity in EEG recorded from a normal subject. The alpha activity extracted by these components were either augmented (*left*) or blocked (*middle*) by subject responses. When the spatial filter for the second alpha component (*middle*) was applied to EEG records from another session in which the subject was asked only to 'mentally note' the occurrence of target stimuli, blocking was replaced by continued phase-locking.

to target stimuli, this alpha activity was not blocked (*right panel*). ICA identifies spatially-overlapping patterns of coherent activity over the entire scalp rather than focusing on single scalp channels or channel pairs.

## 5    Conclusions

We have developed analytic and visualization tools for analysis of multichannel single-trial EEG records. Single-trial ERP analysis based on Independent Component Analysis allows blind separation of multichannel complex EEG data into a sum of temporally independent and spatially fixed components. ICA can effectively remove eye and muscle artifacts without altering the underlying brain activity in the EEG records. ICA can also be used to extract event-related brain phenomena of all types from EEG records. It can identify spatially-overlapping patterns of coherent activity over the entire scalp, and can be used to realign the time courses of response-locked components to prevent temporal smearing in the average arising from performance fluctuations. ERP images make visible systematic relations between single-trial EEG or MEG records and experimental events, and their relations to averaged ERPs. ERP images can also be used to display relationships between phase, amplitude and timing of event-related EEG components time-locked to either stimuli or subject responses. The analysis and visualization tools proposed in this study dramatically increase the amount and quality of information on event- or response-related brain signals that can be extracted from ERP data. Both tools appear applicable to electrophyiological research on normal and clinical populations.

## Footnotes

[1]See [4] for details regarding ICA assumptions underlying EEG analysis.

## References

[1] T.W. Lee, M. Girolami and T.J. Sejnowski (1999) Independent Component Analysis using an Extended Infomax Algorithm for Mixed Sub-Gaussian and Super-Gaussian Sources, *Neural Computation*, **11**(2): 606-33.

[2] H. Yabe, F. Satio & Y. Fukushima (1993) Median Method for Detecting Endogenous Event-related Brain Potentials, *Electroencephalog. clin. Neurophysiolog.* **87**(6):403-7.

[3] H. Yabe, F. Satio & Y. Fukushima (1995) Classification of Single-trial ERP Sub-types: Application of Globally Optimal Vector Quantization Using Simulated Annealing, *Electroencephalog. clin. Neurophysiolog.* **94**(4):288-97.

[4] S. Makeig, T-P Jung, A.J. Bell, D. Ghahremani, and T.J. Sejnowski (1997) Blind Separation of Event-related Brain Responses into Independent Components, *Proc. Natl. Acad. Sci. USA*, USA, **94**:10979-84.

[5] A.J. Bell & T.J. Sejnowski (1995). An information-maximization approach to blind separation and blind deconvolution, *Neural Computation* **7**:1129-1159.

[6] S. Makeig, M. Westerfield, J. Covington, T-P Jung, J. Townsend, T.J. Sejnowski, and E. Courchesne (in press) Functionally independent components of the late positive event-related potential in a visual spatial attention paradigm, *J. Neuroscience*.

[7] Jung T-P, Humphries C, Lee TW, Makeig S, McKeown MJ, Iragui V, Sejnowski TJ (1998) Extended ICA Removes Artifacts from Electroencephalographic Data, In: *Advances in Neural Information Processing Systems* **10**, 894-900.

[8] J.G. Small (1971) Sensory Evoked Responses of Autistic Children, In: *Infantile Autism*, 224-39.
